# Accelerating Bayesian Structural Inference for Non-Decomposable Gaussian Graphical Models

**Baback Moghaddam**
Jet Propulsion Laboratory
California Institute of Technology
baback@jpl.nasa.gov

**Benjamin M. Marlin**
Department of Computer Science
University of British Columbia
bmarlin@cs.ubc.ca

**Mohammad Emtiyaz Khan**
Department of Computer Science
University of British Columbia
emtiyaz@cs.ubc.ca

**Kevin P. Murphy**
Department of Computer Science
University of British Columbia
murphyk@cs.ubc.ca

## Abstract

We make several contributions in accelerating approximate Bayesian structural inference for non-decomposable GGMs. Our first contribution is to show how to efficiently compute a BIC or Laplace approximation to the marginal likelihood of non-decomposable graphs using convex methods for precision matrix estimation. This optimization technique can be used as a fast scoring function inside standard Stochastic Local Search (SLS) for generating posterior samples. Our second contribution is a novel framework for efficiently generating large sets of high-quality graph topologies without performing local search. This graph proposal method, which we call "Neighborhood Fusion" (NF), samples candidate Markov blankets at each node using sparse regression techniques. Our third contribution is a hybrid method combining the complementary strengths of NF and SLS. Experimental results in structural recovery and prediction tasks demonstrate that NF and hybrid NF/SLS out-perform state-of-the-art local search methods, on both synthetic and real-world datasets, when realistic computational limits are imposed.

## 1 Introduction

There are two main reasons to learn the structure of graphical models: knowledge discovery (to interpret the learned topology) and density estimation (to compute log-likelihoods and make predictions). The main difficulty in graphical model structure learning is that the hypothesis space is extremely large, containing up to $2^{d(d-1)/2}$ graphs on $d$ nodes. When the sample size $n$ is small, there can be significant uncertainty with respect to the graph structure. It is therefore advantageous to adopt a Bayesian approach and maintain an approximate posterior over graphs instead of using a single "best" graph, especially since Bayesian model averaging (BMA) can improve predictions.

There has been much work on Bayesian inference for directed acyclic graphical model (DAG) structure, mostly based on Markov chain Monte Carlo (MCMC) or stochastic local search (SLS) [22, 19, 16, 14]. MCMC and SLS methods for DAGs exploit the important fact that the marginal likelihood of a DAG, or an approximation such as the Bayesian Information Criterion (BIC) score, can be computed very efficiently under standard assumptions including independent conjugate priors, and complete data. An equally important property in the DAG setting is that the score can be quickly updated when small local changes are made to the graph. This conveniently allows one to move rapidly through the very large graph space of DAGs.

However, for knowledge discovery, a DAG may be an unsuitable representation for several reasons. First, it does not allow directed cycles, which may be an unnatural restriction in certain domains.

Second, DAGs can only be identified up to Markov equivalence in the general case. In contrast, undirected graphs (UGs) avoid these issues and may be a more natural representation for some problems. Also, for UGs there are fast methods available for identifying the local connectivity at each node (the node's Markov blanket). We note that while the UG and DAG representations have different properties and enable different inference and structure learning algorithms, the distinction between UGs and DAGs from a density estimation perspective may be less important [12].

Most prior work on Bayesian inference for Gaussian Graphical Models (GGMs) has focused on the special case of decomposable graphs (*e.g.*, [17, 2, 29]). The popularity of decomposable GGMs is mostly due to the fact that one can compute the marginal likelihood in closed form using similar assumptions to the DAG case. In addition, one can update the marginal likelihood in constant time after single-edge moves in graph space [17]. However, the space of decomposable graphs is much smaller than the space of general undirected graphs. For example, the number of decomposable graphs on $d$ nodes for $d = 2, \ldots, 8$ is $2, 8, 61, 822, 18154, 617675, 30888596$ [1, p.158]. If we divide the number of decomposable graphs by the number of general undirected graphs, we get the "volume" ratios: $1, 1, 0.95, 0.80, 0.55, 0.29, 0.12$. This means that decomposability significantly limits the subclass of UGs available for modeling purposes, even for small $d$. Several authors have studied Bayesian inference for GGM structure in the general case using approximations to the marginal likelihood based on Monte Carlo methods (*e.g.,* [8, 31, 20, 3]). However, these methods cannot scale to large graphs because of the high computational cost of Monte Carlo approximation.

In this paper, we propose several techniques to help accelerate approximate Bayesian structural inference for non-decomposable GGMs. In Section 2, we show how to efficiently compute BIC and Laplace approximations to the marginal likelihood $p(\mathcal{D}|G)$ by using recent convex optimization methods for estimating the precision matrix of a GGM. In Section 3, we present a novel framework for generating large sets of high-quality graphs which we call "Neighborhood Fusion" (NF). This framework is quite general in scope and can use any Markov blanket finding method to devise a set of probability distributions (proposal densities) over the local topology at each node. It then specifies rules for "fusing" these local densities (via sampling) into an approximate posterior over whole graphs $p(G|\mathcal{D})$. In Section 4, we combine the complementary strengths of NF and existing SLS methods to obtain even higher quality posterior distributions in certain cases. In Section 5, we present an empirical evaluation of both knowledge discovery and predictive performance of our methods. For knowledge discovery, we measure structural recovery in terms of accuracy of finding true edges in synthetic GGMs (with known structure). For predictive performance, we evaluate test set log-likelihood as well as missing-data imputation on real data (with unknown structure). We show that the proposed NF and hybrid NF/SLS methods for general graphs outperform current approaches to GGM learning for both decomposable and general (non-decomposable) graphs.

Throughout this paper we will view the marginal likelihood $p(\mathcal{D}|G)$ as the key to structural inference and as being equivalent to the graph posterior $p(G|\mathcal{D})$ by adopting a *flat* structural prior $p(G)$ w.l.o.g.

## 2    Marginal Likelihood for General Graphs

In this section we will review the G-Wishart distribution and discuss approximations to the marginal likelihood of a non-decomposable GGM under the G-Wishart prior. Unlike the decomposable case, here the marginal likelihood can not be found in closed form. Our main contribution is the insight that recently proposed convex optimization methods for precision matrix estimation can be used to efficiently find the mode of a G-Wishart distribution, which in turn allows for more efficient computation of BIC and Laplace modal approximations to the marginal likelihood.

We begin with some notation. We define $n$ to be the number of data cases and $d$ to be the number of data dimensions. We denote the $i^{th}$ data case by $x_i$ and a complete data set $\mathcal{D}$ with the $n \times d$ matrix $X$, with the corresponding scatter matrix $S = X^T X$ (we assume centered data). We use $G$ to denote an undirected graph, or more precisely its adjacency matrix. Graph edges are denoted by unordered pairs $(i, j)$ and the edge $(i, j)$ is in the graph $G$ if $G_{ij} = 1$. The space of all positive definite matrices having the same zero-pattern as $G$ is denoted by $\mathcal{S}_G^{++}$. The covariance matrix is denoted by $\Sigma$ and its inverse or the precision matrix by $\Omega = \Sigma^{-1}$. We also define $\langle A, B \rangle = \text{Trace}(AB)$.

The Gaussian likelihood $p(\mathcal{D}|\Omega)$ is expressed in terms of the data scatter matrix $S$ in Equation 1. We denote the prior distribution over precision matrices given a graph $G$ by $p(\Omega|G)$. The standard

measure of model quality in the Bayesian model selection setting is the marginal likelihood $p(\mathcal{D}|G)$ which is obtained by integrating $p(\mathcal{D}|\Omega)p(\Omega|G)$ over the space $\mathcal{S}_G^{++}$ as shown in Equation 2.

$$p(\mathcal{D}|\Omega) \;=\; \prod_{i=1}^{n} \mathcal{N}(x_i|\,0, \Omega^{-1}) \;\propto\; |\Omega|^{n/2} \exp(-\frac{1}{2}\langle\Omega, S\rangle) \tag{1}$$

$$p(\mathcal{D}|G) \;=\; \int_{\mathcal{S}_G^{++}} p(\mathcal{D}|\Omega)\, p(\Omega|G)\, d\Omega \tag{2}$$

The G-Wishart density in Equation 3 is the Diaconis-Ylvisaker conjugate form [10] for the GGM likelihood as shown in [27]. The indicator function $I[\Omega \in \mathcal{S}_G^{++}]$ in Equation 3 restricts the density's support to $\mathcal{S}_G^{++}$. The G-Wishart generalizes the hyper inverse Wishart (HIW) distribution to general non-decomposable graphs. The G-Wishart normalization constant $Z$ is shown in Equation 4.

$$W(\Omega|G, \delta_0, S_0) \;=\; \frac{I[\Omega \in \mathcal{S}_G^{++}]}{Z(G, \delta_0, S_0)} \, |\Omega|^{(\delta_0-2)/2} \, \exp(-\frac{1}{2}\langle\Omega, S_0\rangle) \tag{3}$$

$$Z(G, \delta_0, S_0) \;=\; \int_{\mathcal{S}_G^{++}} |\Omega|^{(\delta_0-2)/2} \exp(-\frac{1}{2}\langle\Omega, S_0\rangle)\, d\Omega \tag{4}$$

$$p(\mathcal{D}|G) \;=\; \int_{\mathcal{S}_G^{++}} p(\mathcal{D}|\Omega)\, W(\Omega|G, \delta_0, S_0)\, d\Omega \;\propto\; \frac{Z(G, \delta_n, S_n)}{Z(G, \delta_0, S_0)} \tag{5}$$

Because of the conjugate prior in Equation 3, the $\Omega$ posterior has a similar form $W(\Omega|G, \delta_n, S_n)$ where $\delta_n = \delta_0 + n$ is the posterior degrees of freedom and the posterior scatter matrix $S_n = S + S_0$. The resulting marginal likelihood is then the ratio of the two normalizing terms shown in Equation 5 (which we refer to as $Z_n$ and $Z_0$ for short).

The main drawback of the G-Wishart for general graphs, compared to the HIW for decomposable graphs, is that one cannot compute the normalization terms $Z_n$ and $Z_0$ in closed form. As a result, Bayesian model selection for non-decomposable GGMs relies on approximating the marginal likelihood $p(\mathcal{D}|G)$. The existing literature focuses on Monte Carlo and Laplace approximations. One strategy that makes use of Monte Carlo estimates of both $Z_n$ and $Z_0$ is given by [3]. However, the computation time required to find accurate estimates can be extremely high [20] (see Section 6).

An effective approximation strategy based on using a Laplace approximation to $Z_n$ and a Monte Carlo approximation to $Z_0$ is given in [21]. This requires finding the mode of the G-Wishart, with which a closed-form expression for the Hessian is derived [21]. We consider a simpler method which applies the Laplace approximation to both $Z_n$ and $Z_0$ for greater speed, which we call *full-Laplace*. Nevertheless, computing the Hessian determinant has a computational complexity of $O(E^3)$, where $E$ is the number of edges in $G$. Since $E = O(d^2)$ in the worst-case scenario, computing a full Hessian determinant becomes infeasible for large $d$ in all but the sparsest of graphs.

Due to the high computational cost of Monte Carlo and Laplace approximation in high dimensions, we consider two alternative marginal likelihood approximations that are significantly more efficient. The first alternative is to approximate $Z_n$ and $Z_0$ by Laplace computations in which the Hessian matrix is replaced by its diagonal (by setting off-diagonal elements to zero). We refer to this method as the *diagonal-Laplace* score. The other alternative is the Bayesian Information Criterion (BIC) score shown in Equation 6, which is another large-sample Laplace approximation

$$\text{BIC}(G) \;=\; \log p(\mathcal{D}|\hat{\Omega}_G) - \frac{1}{2} \operatorname{dof}(G) \log n \;, \qquad \operatorname{dof}(G) = d + \sum_{i<j} G_{ij} \tag{6}$$

where, by analogy to [34], we define the GGM's degrees-of-freedom (dof) to be the number of free parameters in the precision matrix. For BIC we use the G-Wishart posterior mode $\hat{\Omega}_G$ as the plug-in estimate, since the MLE is undefined for $n < d$. But we use a vague and proper prior ($\delta_0 = 3$).

Therefore, all three approximations will require finding the mode of a G-Wishart (for the posterior and/or the prior). In [21] an Iterative Proportional Scaling (IPS) algorithm [30] is proposed to find the G-Wishart mode. However, IPS requires finding the maximal cliques of the graph, which is an NP-hard problem. We will now derive a much more efficient G-Wishart mode-finder using convex optimization techniques. We apply this method to find $\hat{\Omega}_G$ when computing BIC scores, as well as the prior and posterior G-Wishart modes when computing Laplace approximations to $Z_0$ and $Z_n$.

Observe that we can express the mode of any G-Wishart distribution with the optimization problem in Equation 7, where the density is parameterized by graph $G$, degree $\delta$ and the scatter matrix $S$.

$$\hat{\Omega}_G \;=\; \arg\max_{\Omega \in \mathcal{S}_G^{++}} \log W(\Omega|G,\delta,S) \;=\; \arg\min_{\Omega \in \mathcal{S}_G^{++}} -\log|\Omega| \;+\; \left\langle \Omega, \frac{S}{\delta-2} \right\rangle \qquad (7)$$

This "COVSEL" type problem [9] is equivalent to finding the maximum likelihood precision matrix of a GGM with known structure $G$, and is a convex optimization problem. Several new methods for solving this precision estimation problem have been recently proposed, and unlike IPS they do not require computing the clique structure of the underlying graph. Hastie *et al.* [18] present one such method which consists of iteratively solving a series of least square problems on the free elements of the precision matrix, which has $O(\sum_i^d (\sum_{j\neq i} G_{ij})^3)$ complexity per iteration [18, p.634].

The G-Wishart mode in Equation 7 can also be found more directly with a gradient-based optimizer such as L-BFGS [6], by using the implementation convention that the objective function is $\infty$ for a non-positive definite matrix. This technique has been used previously by Duchi *et al.* for the more difficult problem of $\ell_1$ penalized precision matrix estimation [13]. The gradient of the objective function is simply set to $(-\Omega^{-1} + S) \odot G$, where $\odot$ indicates element-wise multiplication. The elements of the precision matrix corresponding to absent edges in $G$ are fixed to zero, and we optimize over the remaining elements. The complexity per iteration is $O(d^3)$. In practice, initializing the above optimization with the output of few iterations of the block coordinate descent method of [18] (Glasso with known $G$) is quite effective, as it requires fewer subsequent L-BFGS steps.

In Section 5 we explore the speed *vs.* accuracy trade-off of the various marginal likelihood approximation schemes discussed above; comparing full-Laplace, diagonal-Laplace and the BIC score functions to the marginal likelihood values obtained with the Monte Carlo method of [3].

# 3   Neighborhood Fusion

In this section we describe a novel framework we call "Neighborhood Fusion" (NF) for generating an approximate posterior distribution $p(G|\mathcal{D})$ over general graphs. An important advantage of working with general graphs, instead of decomposable graphs, is that we can leverage simple and stable methods for quickly exploring Markov blankets. One popular method for structural recovery is Glasso which imposes an $l_1$ penalty on $\Omega$ [4, 15, 32]. Finding the corresponding graph takes $O(d^3)$ time per iteration for each setting of the regularization parameter $\lambda$. However, the choice of the $\lambda$ parameter is critical, and in practice we often find that no setting of this parameter leads to good recovery. A related approach, proposed in [23], uses $l_1$-regularized linear regression or Lasso to identify the Markov blanket (MB) of each node. These Markov blankets are then combined using intersection or union (AND/OR) to give the global graph $G$.

These methods essentially produce a single "best" graph, but our main interest is in approximating the full posterior $p(G|\mathcal{D})$. Our NF framework uses a Markov blanket finding method to derive a set of probability distributions over the local topology at each node, and specifies a rule for combining these into an approximate posterior over graphs. The detailed steps of the generic NF algorithm are:

1. Regress each node $i$ on all others to find neighborhoods of all cardinalities $k = 0 : d - 1$ using a sparse regression method. Denote the set of Markov blankets for node $i$ by $\mathcal{N}_i$

2. Compute the linear regression scores $s(b)$ for each Markov blanket $b$ in $\mathcal{N}_i$, and define $p_i(b) = \exp(s(b))/(\sum_{b' \in \mathcal{N}_i} \exp(s(b')))$ as the node's Markov blanket *proposal* density

3. Independently sample a Markov blanket for each node $i$ from its proposal density $p_i(b)$, and then combine all $d$ sampled Markov blankets to assemble a single graph $G$

4. Find $G$'s precision matrix using Equation 7 and compute the graph score as in Section 2

5. Repeat sampling step 3 and 4 to produce a large ensemble of posterior-weighted graphs

The design choices in the NF framework are the choice of a sparse linear regression method (and its score function), the choice of a method for combining Markov blankets, and the choice of a graph score function (for marginal likelihood). In all the results that follow we use the linear regression BIC score induced by regressing node $i$ on $\mathcal{N}_i$, and generate whole graphs by intersecting the

Markov blankets using the AND operator. This essentially constitutes sampling from the "AND-censored" *pseudo* marginal likelihood and is therefore likely to produce good candidate MBs that can be fused into high-quality graphs. Note that the uncertainty modeled by the MB proposal density is critical, as it promotes efficient exploration of model space to generate a large variety of high-scoring models. Indeed, the best NF-sampled graphs typically have higher scores than the *pseudo* "MAP" graph obtained by simply intersecting the best MBs [23], due to the inherent noise in the linear regression BIC scores and the possibility of over-fitting. Moreover, our MB proposals can be "flattened" with a temperature parameter to trade-off exploration *vs.* fidelity of the sampled graphs, though we generally find it unnecessary to go to such extremes and use a default temperature of one.

We next consider two further specialized instances of the NF framework using different sparse linear regression methods. The first method uses the full Lasso/LARS regularization path and is called L1MB ("L1 Markov Blanket") which we adapted from the DAG-learning method of [28]. NF based on these $l_1$-derived MBs we call NF-L1MB (or NF-L1 for short). In light of recent theoretical results on the superiority of greedy forward/backward search over Lasso [33] we also use the $l_0$-based method of [24] which we call L0MB ("L0 Markov blanket"). And NF based on L0MB we will call NF-L0MB (or NF-L0 for short). Our experimental results show that the improvement of the $l_0$-based greedy search of [24] over Lasso/LARS translates directly to obtaining improved MB proposals with NF-L0MB compared to NF-L1MB. Similar forward/backward greedy variable selection techniques were put to good use in the "compositional network" DAG-to-UG method of [11], however not for deriving proposal distributions for parents/MBs as we do here for NF.

Our overall computational scheme is quite fast by design: finding MB proposals is at most $O(d^4)$ with L1MB/L0MB (although L0MB has a smaller constant for both the forward and backward passes). Thereafter, we sample full graphs in $O(d^2)$ time (since we are sampling a discrete p.m.f. for $d$ MB candidates at each node) and computing a G-Wishart mode $\hat{\Omega}_G$ is just $O(d^3)$ per iteration.

# 4    Stochastic Local Search

Stochastic Local Search (SLS) can also be viewed as a mechanism for generating an approximate posterior distribution over graphs. Like MCMC methods, SLS explores high probability regions of graph space, but unlike MCMC it computes approximate model probabilities directly for each graph it visits. This is sensible for large discrete hypothesis spaces like the space of UGs since the chance of visiting the same graph multiple times is extremely small. We note that SLS represents an orthogonal and complementary approach to structural inference relative to the NF framework presented in Section 3. In this section we discuss SLS for both decomposable and general (non-decomposable) GGMs. Specifically, we describe new initialization and edge-marginal updating methods for non-decomposable GGMs, and also introduce a highly effective hybrid NF/SLS method.

SLS with decomposable graphs has the advantage that its natural scoring function, the marginal likelihood, can be computed exactly under the conjugate Hyper Inverse Wishart prior. The marginal likelihood can also be updated efficiently when local changes are made to the underlying graph. A state-of-the-art SLS method for decomposable GGMs is given in [29], which can be used with an arbitrary score function over the space of general graphs. Here we consider SLS for general graphs using the Laplace score described in Section 2. In the SLS in [29], at iteration $t$, an edge $(i, j)$ from $G^t$ is chosen at random and flipped with probability $q_{ij}$. If the resulting graph is admissible and has not been visited before, this graph becomes $G^{t+1}$, and we evaluate its score. In the general case, every new graph generated is admissible. In the decomposable case, only decomposable graphs are admissible. We should note that unlike exhaustive search methods, this method avoids evaluating the score of all $O(d^2)$ neighboring graphs at each iteration, and instead picks one at random.

There are two key modifications used in [29] which help this method work well in practice. First, the marginal edge probabilities $q_{ij}$ are updated online, so edges that have proved useful in the past are more likely to be proposed in the future. Second, on each iteration the algorithm chooses to perform a resampling step with probability $p_r$ or a global move with probability $p_g$. In a resampling step we set $G^{t+1}$ to $G^v$, where $v \leq t$, with probability proportional to the score (or exponentiated score) of $G^v$. In a global move we sample a completely new graph (based on the edge marginals $q_{ij}$) for $G^{t+1}$. We note that a similar idea of using edge-marginals to propose moves in DAG space was suggested in [14]. In this paper, we set $p_r = 0.02$ and $p_g = 0$ (*i.e.*, we do not use global moves).

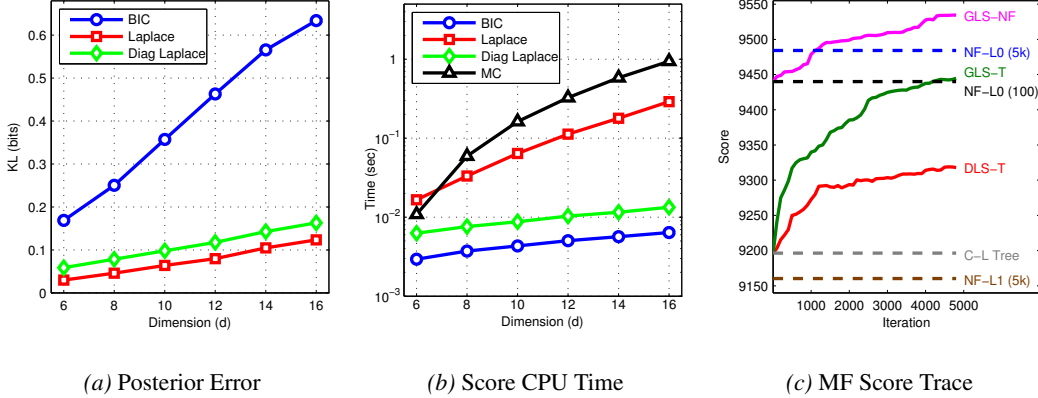

*(a)* Posterior Error       *(b)* Score CPU Time       *(c)* MF Score Trace

*Figure 1:* Score trade-offs: (a) average KL error of posterior approximations and (b) the average time to score a single graph as a function of data dimensionality. (c) Results on the MF dataset: scores for various methods.

We now propose a new initialization and updating scheme for non-decomposable SLS based on a set of $k$ initial graphs $G_0^1, ..., G_0^k$ (with positive weights $w_0^1, ..., w_0^k$ defined by normalized scores) obtained from our NF graph-sampling framework. Our approach views $q_{ij}$ as a Beta random variable with prior parameters $\alpha_{ij} = \sum_{l=1}^k w_0^l G_{0,ij}^l$ and $\beta_{ij} = \sum_{l=1}^k w_0^l (1 - G_{0,ij}^l)$. We update this distri-bution online using $p(q_{ij}|G^{1:t}) = \text{Beta}(\alpha_{ij} + t f_{ij}^t, \beta_{ij} + t(1 - f_{ij}^t))$, where $f_{ij}^t = \frac{\sum_{l=1}^t G_{ij}^l \ p(D|G^l)}{\sum_{l=1}^t p(D|G^l)}$. We then flip an edge with probability $E[q_{ij}] = (\alpha_{ij} + t f_{ij}^t)/(\alpha_{ij} + \beta_{ij} + t)$.

SLS's main drawback is that, if started from the empty graph as in [29], it will necessarily take at least $E$ steps to find the highest scoring graph, where $E$ is the number of true edges. This means that it will likely require a very large number of iterations even in moderately large and dense graphs. An improved initialization strategy is to start the search from the optimal tree, which can be found in $O(d^2)$ time using the Chow-Liu algorithm [7]. An even better initialization strategy, for non-decomposable graphs, is to "seed" SLS with a batch of NF-sampled graphs for $G_0^1, ..., G_0^k$ and then start the search by executing a resampling step. In this way, a limited number of SLS steps can effectively explore the space around these initial high-quality graphs. We refer to this new method, where NF is used to both initialize the edge-marginals and seed the graph history, as *hybrid NF/SLS*.

# 5 Experiments

We begin our experimental analysis by first assessing the speed *vs.* accuracy trade-off of the different marginal likelihood approximations in Section 2. For this evaluation we use the Monte Carlo method of [3] as a proxy for the ground truth marginal likelihood. For data dimensions $d = 6, ..., 16$, we sample 100 random, sparse precision matrices with an average edge density of 0.5. For each sampled precision matrix $\Omega$ we generate $10d$ observations from the corresponding GGM. Using each approximation method, we score all $d(d-1)/2$ neighbors of $G$ obtained from $G$ by single edge flips. We then compute a posterior distribution over this set of graphs by normalizing the scores (or exponentiated scores as appropriate). We then compute the Kullback-Leibler (KL) divergence from the Monte Carlo based posterior to each approximate posterior. We also record the time required to score each graph. The scoring methods we use are BIC, full-Laplace and diagonal-Laplace for $Z_n$ and $Z_0$. We use a G-Wishart prior with parameters $\delta_0 = 3$ and $S_0 = I$. In Figure 1(a) we show the average error of these posterior approximations as a function of data dimensionality $d$, as measured by KL divergence. In Figure 1(b) we show the average time required to score a single graph as a function of graph size $d$. As expected, full-Laplace is the most accurate and most costly of the approximations next to Monte Carlo. Interestingly, diagonal-Laplace appears to be significantly more accurate than BIC (for this test) and is in fact only twice as costly. Moreover, diagonal-Laplace is already more than 20 times faster than Monte Carlo and full-Laplace at $d = 16$. On the basis of the speed *vs.* accuracy trade-off seen in Figure 1(a) and Figure 1(b), we will report only the diagonal-Laplace score in the remainder of our experiments.

We next evaluate the NF-L1MB and NF-L0MB methods described in Section 3 (note that we will use the short labels NF-L1 and NF-L0 in the Figures), and SLS for decomposable and general graphs

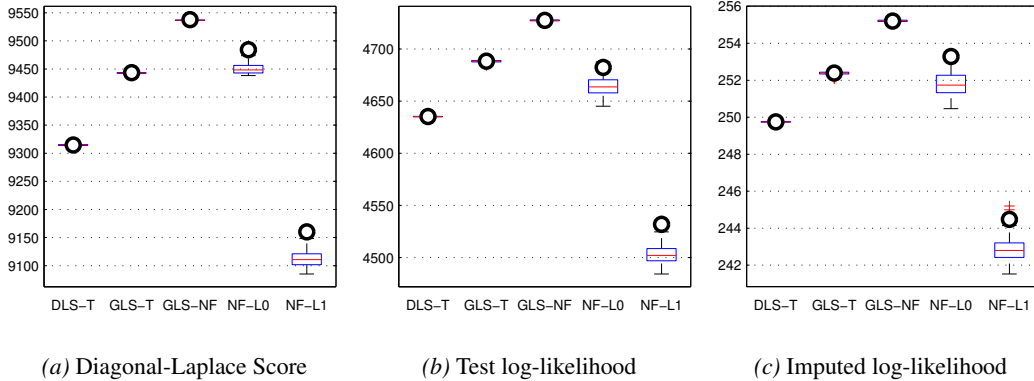

*Figure 2:* Mutual Fund results: box plots of the (a) scores, (b) test set log-likelihoods and (c) test set imputation log-likelihoods (averaged over all possible missing 3-nodes). The BMA performance is indicated with a circle.

initialized from the optimal tree as described in Section 4 (denoted as DLS-T and GLS-T, respectively), and a L0MB-based hybrid NF/SLS method as described in Section 4 (denoted as GLS-NF). We sample 5000 graphs for each of the NF methods and run each of the SLS methods for 5000 steps, also producing 5000 graphs. The hybrid NF/SLS method is initialized with a sample of 100 NF graphs, and then run for 5000 steps. We compute the score for each set of graphs (diagonal-Laplace for non-decomposable and exact marginal likelihood for decomposables). We extract the 100 best graphs by score, and produce an approximation to $p(G|\mathcal{D})$ by normalizing the exponentiated scores. We report results for individual graphs in the best 100, but our main focus is on performance statistics under Bayesian model averaging (BMA) with approximate scores of each method. In the following experiments we use a G-Wishart prior degree $\delta_0 = 3$ (the smallest integer yielding a proper prior) and unless otherwise noted, a default prior scatter matrix of $S_0 = \text{mean}(\text{diag}(\text{cov}(X))) \cdot I_d$.

We examine the two main inferential tasks of prediction and knowledge discovery. We first measure the predictive ability of each method by computing both test set log-likelihoods and test set imputation log-likelihoods. For this task we use the "Mutual Funds" (MF) dataset used by [29] for SLS with decomposable GGMs, with $d = 59$, which they split into 60 months of training data and 26 months of test data. But due to the resulting critical sampling ($n \approx d$), here we use a more stable $S_0 = \rho \cdot \text{Diag}(X^T X)$ with $\rho = 0.055$ (a Ledoit-Wolf shrinkage). In Figure 1(c) we show a trace plot of scores for the SLS methods and best scores for the NF and tree methods. Box plots of diagonal-Laplace scores for each method on the MF data are shown in Figure 2(a). The corresponding test set log-likelihoods are shown in Figure 2(b). For the imputation experiment, we impute "missing" triplets of variables given the values of the remaining variables. We compute the log-likelihood of this predictive (imputed) distribution by averaging it over all 59-choose-3 = 32509 possible missing patterns and all 26 test cases. The imputation log-likelihoods are shown in Figure 2(c). We can see that NF-L0MB out-performs NF-L1MB on both predictive tasks (full and missing). Interestingly, on this small data set SLS for general graphs (GLS-T) performs rather well. But our hybrid NF-L0MB "seeding" approach for SLS (GLS-NF) has the best overall BMA performance.

In the second set of tasks, we evaluate the structural recovery of each method by measuring the true positive and false positive rates for edge inclusion w.r.t. a ground-truth GGM. The synthetic data sets contain $d = 100$ nodes, $E = 300$ edges and $n/d$ ratios of 5.0 (Synth-1) and 0.5 (Synth-2). Synth-1 is thus generously oversampled while Synth-2 is undersampled. Both synthetic GGMs were generated by moralizing a *random* DAG. Figures 3(a) and 3(b) show plots of TPR *vs.* FPR for edge recovery. The rates for individual graphs are shown as small grey symbols while the BMA rate is shown with a large bold colored symbol. The results show that NF-L0MB and GLS-NF (based on seeding GLS with 100 NF-L0MB graphs) are the best methods on both data sets. We also see that NF-L0MB dominates NF-L1MB, while the hybrid GLS-NF dominates both GLS-T and DLS-T.

For the $d = 59$ MF dataset in Figure 1(c), NF-sampling 5000 graphs and doing the G-Wishart mode-fits and diagonal-Laplace scoring, takes a total of 13 mins, and likewise 30 mins for the synthetic $d = 100$ dataset in Figure 3. Generating and scoring 5000 graphs with non-decomposable SLS takes 37 mins on the MF dataset and 59 mins on the synthetic one. Decomposable SLS takes 31 mins on MF and 43 mins on the synthetic. All times quoted are for Matlab code running on a 3.16 GHz PC.

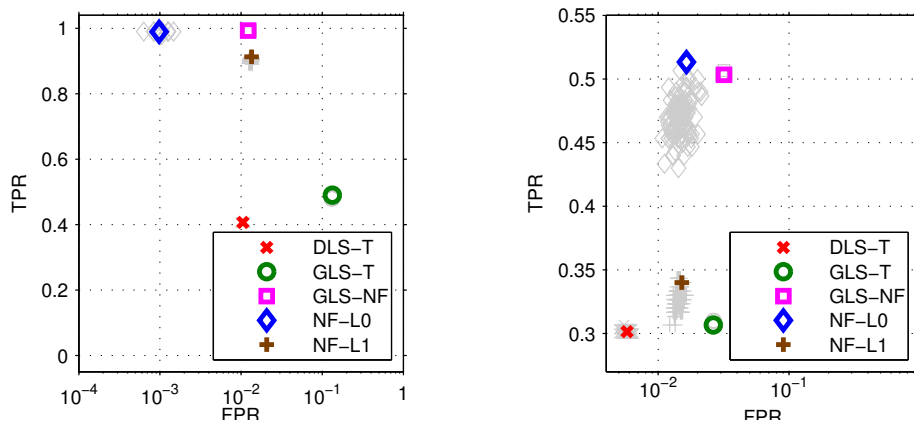

*(a)* Synth-1: $n/d = 5.0$  $(d = 100)$          *(b)* Synth-2: $n/d = 0.5$  $(d = 100)$

*Figure 3:* True Positive *vs.* False Positive rates for (a) Synth-1 and (b) Synth-2 datasets for each recovery method. The top 100 graphs are shown with a grey symbol and the bold colored symbol is the BMA graph.

## 6   Discussion

We offer a practical framework for fast inference in non-decomposable GGMs providing reasonable accuracy for marginal likelihoods. While Monte Carlo methods are the "gold standard" (modulo the usual convergence issues) they are exorbitantly costly for even moderately large $d$. For example, scoring all the neighbors of a 150-node graph via SLS required over 40 days of computation in [20]. A similar size task would take less than 40 mins with our diagonal-Laplace approximation method.

As pointed out by [21] there may not always be sufficient concentration for a Laplace approximation to $Z_0$ to be very accurate, which is why they use MC for this quantity. We chose Laplace for both $Z_n$ and $Z_0$ solely for speed (to avoid MC altogether) and found good agreement between full-Laplace and BIC for much larger graphs than in Figure 1(a). Our Laplace scores also roughly matched the MC values for the Fisher Iris data in [3], selecting essentially the same top-ranked 16 graphs (see Figure 5 in [3]). Using a diagonal instead of a full Hessian was yet another compromise for speed. An issue that should be explored further is the sensitivity of these approximations to different priors.

We experimentally validated NF on nearly $10^4$ synthetic cases ranging in size from $d = 10, ..., 500$, with various edge densities and $n/d$ ratios, with consistently good results, typified by the two test cases shown in Figure 3. Note that the sub-par performance of NF-L1 is not a failing of NF but due to $l_1$-based MBs, and superiority of $l_0$-based F/B greedy search is not without precedent [25, 24, 33]. We note that NF can be partially justified as a *pseudo* marginal likelihood (PML), but whereas most authors rely only on its maximizer [23] we exploit the full (pseudo) density. Without the AND filter, NF-drawn MBs are sampled from a set of "consistent" full-conditionals in the sense of Besag [5], and their max-BIC MBs are collectively the PML mode (note that here we mean the node regression BIC, not graph BIC). Enforcing AND is a necessary domain truncation for a valid UG which alters the mode. This symmetrized "pseudo-MAP" $G$ is often an average-scoring one compared to the best and worst found by NF, which motivates BMA and justifies NF. We can also view NF as an *over-dispersed* proposal density; its weighted graphs a rough proxy for $p(G|\mathcal{D})$. This approximation may be biased but our results show it is quite useful for prediction and imputation (and seeding SLS with high-quality graphs). Finally, while use of BIC/Laplace for hypothesis testing is often criticized, it can still be useful for *estimation* [26], and nowhere in our framework are these scores being used to select a single "best" model (whether it be a MB or a $G$) due to our reliance on sampling and BMA.

### Acknowledgments

We like to thank the reviewers for their helpful and encouraging feedback. *BMM* was supported by the Killam Trusts at UBC and *KPM* would like to thank NSERC and CIFAR. This work was in part carried out at the Jet Propulsion Laboratory, California Institute of Technology, under a contract with the National Aeronautics and Space Administration.

# References

[1] H. Armstrong. *Bayesian Estimation of Decomposable GGMs*. PhD thesis, UNSW, 2005.

[2] H. Armstrong, C. Carter, K. Wong, and R. Kohn. Bayesian covariance matrix estimation using a mixture of decomposable graphical models. *Statistics and Computing*, 2008.

[3] A. Atay-Kayis and H. Massam. A Monte Carlo method for computing the marginal likelihood in non-decomposable Gaussian graphical models. *Biometrika*, 92, 2005.

[4] O. Banerjee, L. El Ghaoui, A. d'Aspremont, and G. Natsoulis. Convex optimization techniques for fitting sparse Gaussian graphical models. In *Intl. Conf. on Machine Learning*, 2006.

[5] J. Besag. Efficiency of pseudo-likelihood estimation for simple Gaussian fields. *Biometrika*, 1977.

[6] R. Byrd, P. Lu, J. Nocedal, and C. Zhu. A limited memory algorithm for bound constrained optimization. *SIAM J. of Scientific & Statistical Computing*, 16(5), 1995.

[7] C. Chow and C. Liu. Approximating discrete probability distributions with dependence trees. *IEEE Trans. on Info. Theory*, 14, 1968.

[8] P. Dellaportas, P. Giudici, and G. Roberts. Bayesian inference for nondecomposable graphical Gaussian models. *Sankhya, Ser. A*, 65, 2003.

[9] A. Dempster. Covariance selection. *Biometrics*, 28(1), 1972.

[10] P. Diaconis and D. Ylvisaker. Conjugate priors for exponential families. *Annals of statistics*, 7(2), 1979.

[11] D. Dobra, C. Hans, B. Jones, J. Nevins, G. Yao, and M. West. Sparse graphical models for exploring gene expression data. *J. Multivariate analysis*, 90, 2004.

[12] J. Domke, A. Karapurkar, and Y. Aloimonos. Who killed the directed model? In *CVPR*, 2008.

[13] J. Duchi, S. Gould, and D. Koller. Projected subgradients for learning sparse Gaussians. In *UAI*, 2008.

[14] D. Eaton and K. Murphy. Bayesian structure learning using DP and MCMC. In *UAI*, 2007.

[15] J. Friedman, T. Hastie, and R. Tibshirani. Sparse inverse covariance estimation in Glasso. *Biostats*, 2007.

[16] N. Friedman and D. Koller. Being Bayesian about network structure: A Bayesian approach to structure discovery in Bayesian networks. *Machine Learning*, 50, 2003.

[17] P. Giudici and P. Green. Decomposable graphical Gaussian model determination. *Biometrika*, 1999.

[18] T. Hastie, R. Tibshirani, and J. Friedman. *The Elements of Statistical Learning*. Springer, 2009.

[19] D. Heckerman, D. Geiger, and M. Chickering. Learning Bayesian networks: the combination of knowledge and statistical data. *Machine Learning*, 20(3), 1995.

[20] B. Jones, C. Carvalho, A. Dobra, C. Hans, C. Carter, and M. West. Experiments in stochastic computation for high-dimensional graphical models. *Statistical Science*, 20, 2005.

[21] A. Lenkoski and A. Dobra. Bayesian structural learning and estimation in Gaussian graphical models. Technical Report 545, Department of Statistics, University of Washington, 2008.

[22] D. Madigan and A. Raftery. Model selection and accounting for model uncertainty in graphical models using Occam's window. *J. of the Am. Stat. Assoc.*, 89, 1994.

[23] N. Meinshausen and P. Buhlmann. High dimensional graphs and variable selection with the Lasso. *The Annals of Statistics*, 2006.

[24] B. Moghaddam, A. Gruber, Y. Weiss, and S. Avidan. Sparse regression as a sparse eigenvalue problem. In *Information Theory & Applications Workshop (ITA'08)*, 2008.

[25] B. Moghaddam, Y. Weiss, and S. Avidan. Spectral bounds for sparse PCA: Exact & greedy algorithms. In *NIPS*, 2006.

[26] A. Raftery. Bayesian model selection in social research. *Sociological Methodology*, 25, 1995.

[27] A. Roverato. Hyper inverse Wishart distribution for non-decomposable graphs and its application to Bayesian inference for Gaussian graphical models. *Scand. J. Statistics*, 29, 2002.

[28] M. Schmidt, A Niculescu-Mizil, and K Murphy. Learning graphical model structure using $l_1$ regularization paths. In *AAAI*, 2007.

[29] J. Scott and C. Carvalho. Feature-inclusion stochastic search for Gaussian graphical models. *J. of Computational and Graphical Statistics*, 17(4), 2008.

[30] T. Speed and H. Kiiveri. Gaussian Markov distributions over finite graphs. *Annals of Statistics*, 1986.

[31] F. Wong, C. Carter, and R. Kohn. Efficient estimation of covariance selection models. *Biometrika*, 2003.

[32] M. Yuan and Yi Lin. Model selection and estimation in the GGM. *Biometrika*, 94(1), 2007.

[33] T. Zhang. Adaptive forward-backward greedy algorithm for sparse learning. In *NIPS*, 2008.

[34] H. Zou, T. Hastie, and R. Tibshirani. On the "degrees of freedom" of Lasso. *Annals of Statistics*, 2007.
